Teaching Artificial Neural Systems to Drive:
Manual Training Techniques for Autonomous Systems

J. F. Shepanski and S. A. Macy

TRW, Inc.
One Space Park, O2/1779
Redondo Beach, CA 90278

## Abstract

We have developed a methodology for manually training autonomous control systems based on artificial neural systems (ANS). In applications where the rule set governing an expert's decisions is difficult to formulate, ANS can be used to extract rules by associating the information an expert receives with the actions he takes. Properly constructed networks imitate rules of behavior that permits them to function autonomously when they are trained on the spanning set of possible situations. This training can be provided manually, either under the direct supervision of a system trainer, or indirectly using a background mode where the network assimilates training data as the expert performs his day-to-day tasks. To demonstrate these methods we have trained an ANS network to drive a vehicle through simulated freeway traffic.

## Introduction

Computational systems employing fine grained parallelism are revolutionizing the way we approach a number of long standing problems involving pattern recognition and cognitive processing. The field spans a wide variety of computational networks, from constructs emulating neural functions, to more crystalline configurations that resemble systolic arrays. Several titles are used to describe this broad area of research, we use the term artificial neural systems (ANS). Our concern in this work is the use of ANS for manually training certain types of autonomous systems where the desired rules of behavior are difficult to formulate.

Artificial neural systems consist of a number of processing elements interconnected in a weighted, user-specified fashion, the interconnection weights acting as memory for the system. Each processing element calculates an output value based on the weighted sum of its inputs. In addition, the input data is correlated with the output or desired output (specified by an instructive agent) in a training rule that is used to adjust the interconnection weights. In this way the network learns patterns or imitates rules of behavior and decision making.

The particular ANS architecture we use is a variation of Rummelhart et. al. [1] multi-layer perceptron employing the generalized delta rule (GDR). Instead of a single, multi-layer structure, our final network has a a multiple component or "block" configuration where one block's output feeds into another (see Figure 3). The training methodology we have developed is not tied to a particular training rule or architecture and should work well with alternative networks like Grossberg's adaptive resonance model[2].

The equations describing the network are derived and described in detail by Rumelhart et. al.[1]. In summary, they are:

Transfer function: $\quad o_j = (1 + e^{-S_j})^{-1}, \quad S_j = \sum_{i=0}^{n} w_{ji} o_{i};$  (1)

Weight adaptation rule: $\quad \Delta w_{ji} = (1 - \alpha_{ji}) \eta_{ji} \delta_j o_i + \alpha_{ji} \Delta w_{ji}^{\text{previous}};$  (2)

Error calculation: $\quad \delta_j = o_j (1 - o_j) \sum_{k=1}^{m} \delta_k w_{kj},$  (3)

where $o_j$ is the output of processing element $j$ or a sensor input, $w_{ji}$ is the interconnection weight leading from element $i$ to $j$, $n$ is the number of inputs to $j$, $\Delta w$ is the adjustment of $w$, $\eta$ is the training constant, $\alpha$ is the training "momentum," $\delta_j$ is the calculated error for element $j$, and $m$ is the fanout of a given element. Element zero is a constant input, equal to one, so that $w_{j0}$ is equivalent to the bias threshold of element $j$. The $(1 - \alpha)$ factor in equation (2) differs from standard GDR formulation, but it is useful for keeping track of the relative magnitudes of the two terms. For the network's output layer the summation in equation (3) is replaced with the difference between the desired and actual output value of element $j$.

These networks are usually trained by presenting the system with sets of input/output data vectors in cyclic fashion, the entire cycle of database presentation repeated dozens of times. This method is effective when the training agent is a computer operating in batch mode, but would be intolerable for a human instructor. There are two developments that will help real-time human training. The first is a more efficient incorporation of data/response patterns into a network. The second, which we are addressing in this paper, is a suitable environment wherein a man and ANS network can interact in training situation with minimum inconvenience or boredom on the human's part. The ability to systematically train networks in this fashion is extremely useful for developing certain types of expert systems including automatic signal processors, autopilots, robots and other autonomous machines. We report a number of techniques aimed at facilitating this type of training, and we propose a general method for teaching these networks.

## System Development

Our work focuses on the utility of ANS for system control. It began as an application of Barto and Sutton's associative search network[3]. Although their approach was useful in a number of ways, it fell short when we tried to use it for capturing the subtleties of human decision-making. In response we shifted our emphasis from constructing goal functions for automatic learning, to methods for training networks using direct human instruction. An integral part of this is the development of suitable interfaces between humans, networks and the outside world or simulator. In this section we will report various approaches to these ends, and describe a general methodology for manually teaching ANS networks. To demonstrate these techniques we taught a network to drive a robot vehicle down a simulated highway in traffic. This application combines binary decision making and control of continuous parameters.

Initially we investigated the use of automatic learning based on goal functions[3] for training control systems. We trained a network-controlled vehicle to maintain acceptable following distances from cars ahead of it. On a graphics workstation, a one lane circular track was

constructed and occupied by two vehicles: a network-controlled robot car and a pace car that varied its speed at random.. Input data to the network consisted of the separation distance and the speed of the robot vehicle. The values of a goal function were translated into desired output for GDR training. Output controls consisted of three binary decision elements: 1) accelerate one increment of speed, 2) maintain speed, and 3) decelerate one increment of speed. At all times the desired output vector had exactly one of these three elements active. The goal function was quadratic with a minimum corresponding to the optimal following distance. Although it had no direct control over the simulation, the goal function positively or negatively reinforced the system's behavior.

The network was given complete control of the robot vehicle, and the human trainer had no influence except the ability to start and terminate training. This proved unsatisfactory because the initial system behavior--governed by random interconnection weights--was very unstable. The robot tended to run over the car in front of it before significant training occurred. By carefully halting and restarting training we achieved stable system behavior. At first the following distance maintained by the robot car oscillated as if the vehicle was attached by a spring to the pace car. This activity gradually damped. After about one thousand training steps the vehicle maintained the optimal following distance and responded quickly to changes in the pace car's speed.

Constructing composite goal functions to promote more sophisticated abilities proved difficult, even ill-defined, because there were many unspecified parameters. To generate goal functions for these abilities would be similar to conventional programming--the type of labor we want to circumvent using ANS. On the other hand, humans are adept at assessing complex situations and making decisions based on qualitative data, but their "goal functions" are difficult if not impossible to capture analytically. One attraction of ANS is that it can imitate behavior based on these elusive rules without formally specifying them. At this point we turned our efforts to manual training techniques.

The initially trained network was grafted into a larger system and augmented with additional inputs: distance and speed information on nearby pace cars in a second traffic lane, and an output control signal governing lane changes. The original network's ability to maintain a safe following distance was retained intact. This grafting procedure is one of two methods we studied for adding new abilities to an existing system. (The second, which employs a block structure, is described below.) The network remained in direct control of the robot vehicle, but a human trainer instructed it when and when not to change lanes. His commands were interpreted as the desired output and used in the GDR training algorithm. This technique, which we call coaching, proved useful and the network quickly correlated its environmental inputs with the teacher's instructions. The network became adept at changing lanes and weaving through traffic. We found that the network took on the behavior pattern of its trainer. A conservative teacher produced a timid network, while an aggressive trainer produced a network that tended to cut off other automobiles and squeeze through tight openings. Despite its success, the coaching method of training did not solve the problem of initial network instability.

The stability problem was solved by giving the trainer direct control over the simulation. The system configuration (Figure 1), allows the expert to exert control or release it to the network. During initial training the expert is in the driver's seat while the network acts the role of

apprentice. It receives sensor information, predicts system commands, and compares its predictions against the desired output (ie. the trainer's commands). Figure 2 shows the data and command flow in detail. Input data is processed through different channels and presented to the trainer and network. Where visual and audio formats are effective for humans, the network uses information in vector form. This differentiation of data presentation is a limitation of the system; removing it is a task for future research. The trainer issues control commands in accordance with his assigned task while the network takes the trainer's actions as desired system responses and correlates these with the input. We refer to this procedure as master/apprentice training, network training proceeds invisibly in the background as the expert proceeds with his day to day work. It avoids the instability problem because the network is free to make errors without the adverse consequence of throwing the operating environment into disarray.

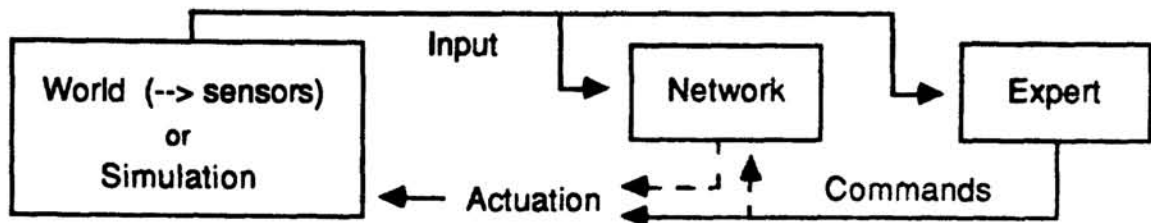

Figure 1. A scheme for manually training ANS networks. Input data is received by both the network and trainer. The trainer issues commands that are actuated (solid command line), or he coaches the network in how it ought to respond (broken command line).

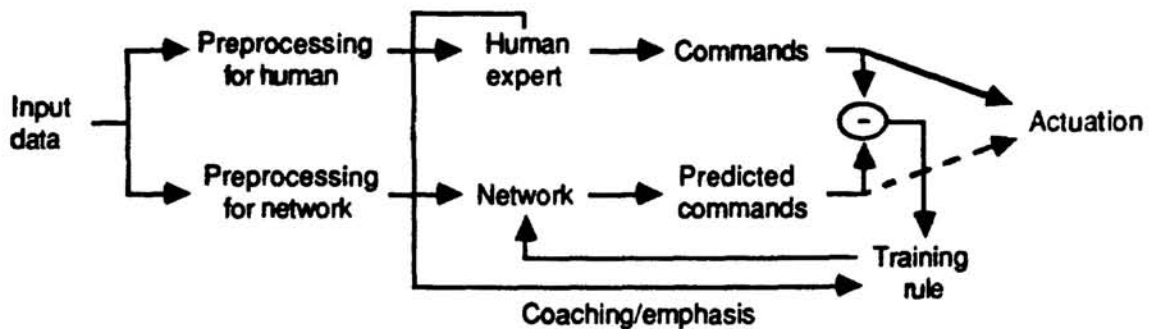

Figure 2. Data and command flow in the training system. Input data is processed and presented to the trainer and network. In master/apprentice training (solid command line), the trainer's orders are actuated and the network treats his commands as the system's desired output. In coaching, the network's predicted commands are actuated (broken command line), and the trainer influences weight adaptation by specifying the desired system output and controlling the values of training constants--his "suggestions" are not directly actuated.

Once initial, background training is complete, the expert proceeds in a more formal manner to teach the network. He releases control of the command system to the network in order to evaluate its behavior and weaknesses. He then resumes control and works through a

series of scenarios designed to train the network out of its bad behavior. By switching back and forth between human and network control, the expert assesses the network's reliability and teaches correct responses as needed. We find master/apprentice training works well for behavior involving continuous functions, like steering. On the other hand, coaching is appropriate for decision functions, like when the car ought to pass. Our methodology employs both techniques.

### The Driving Network

The fully developed freeway simulation consists of a two lane highway that is made of joined straight and curved segments which vary at random in length (and curvature). Several pace cars move at random speeds near the robot vehicle. The network is given the tasks of tracking the road, negotiating curves, returning to the road if placed far afield, maintaining safe distances from the pace cars, and changing lanes when appropriate. Instead of a single multi-layer structure, the network is composed of two blocks; one controls the steering and the other regulates speed and decides when the vehicle should change lanes (Figure 3). The first block receives information about the position and speed of the robot vehicle relative to other cars in its vicinity. Its output is used to determine the automobile's speed and whether the robot should change lanes. The passing signal is converted to a lane assignment based on the car's current lane position. The second block receives the lane assignment and data pertinent to the position and orientation of the vehicle with respect to the road. The output is used to determine the steering angle of the robot car.

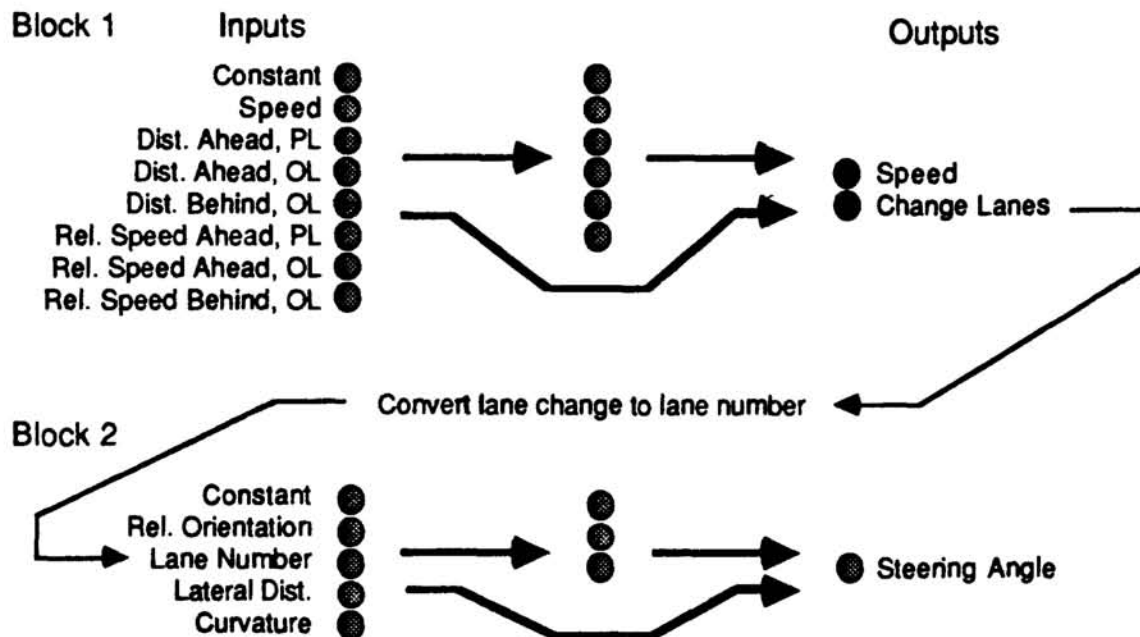

Figure 3. The two blocks of the driving ANS network. Heavy arrows indicate total interconnectivity between layers. PL designates the traffic lane presently occupied by the robot vehicle, OL refers to the other lane, curvature refers to the road, lane number is either 0 or 1, relative orientation and lateral distance refers to the robot car's direction and position relative to the road's direction and center line, respectively.

The input data is displayed in pictorial and textual form to the driving instructor. He views the road and nearby vehicles from the perspective of the driver's seat or overhead. The network receives information in the form of a vector whose elements have been scaled to unitary order, $O(1)$. Wide ranging input parameters, like distance, are compressed using the hyperbolic tangent or logarithmic functions. In each block, the input layer is totally interconnected to both the output and a hidden layer. Our scheme trains in real time, and as we discuss later, it trains more smoothly with a small modification of the training algorithm.

Output is interpreted in two ways: as a binary decision or as a continuously varying parameter. The first simply compares the sigmoid output against a threshold. The second scales the output to an appropriate range for its application. For example, on the steering output element, a 0.5 value is interpreted as a zero steering angle. Left and right turns of varying degrees are initiated when this output is above or below 0.5, respectively.

The network is divided into two blocks that can be trained separately. Beside being conceptually easier to understand, we find this component approach is easy to train systematically. Because each block has a restricted, well-defined set of tasks, the trainer can concentrate specifically on those functions without being concerned that other aspects of the network behavior are deteriorating.

We trained the system from bottom up, first teaching the network to stay on the road, negotiate curves, change lanes, and how to return if the vehicle strayed off the highway. Block 2, responsible for steering, learned these skills in a few minutes using the master/apprentice mode. It tended to steer more slowly than a human but further training progressively improved its responsiveness.

We experimented with different training constants and "momentum" values. Large $\eta$ values, about 1, caused weights to change too coarsely. $\eta$ values an order of magnitude smaller worked well. We found no advantage in using momentum for this method of training, in fact, the system responded about three times more slowly when $\alpha = 0.9$ than when the momentum term was dropped. Our standard training parameters were $\eta = 0.2$, and $\alpha = 0.0$

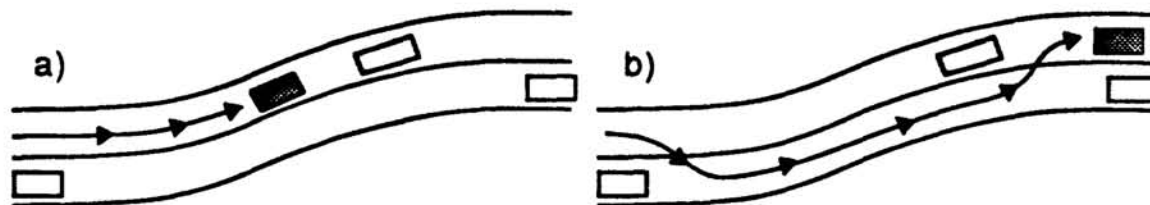

Figure 4. Typical behavior of a network-controlled vehicle (dark rectangle) when trained by a) a conservative driver, and b) a reckless driver. Speed is indicated by the length of the arrows.

After Block 2 was trained, we gave steering control to the network and concentrated on teaching the network to change lanes and adjust speed. Speed control in this case was a continuous variable and was best taught using master/apprentice training. On the other hand, the binary decision to change lanes was best taught by coaching. About ten minutes of training were needed to teach the network to weave through traffic. We found that the network readily adapts the

behavioral pattern of its trainer. A conservative trainer generated a network that hardly ever passed, while an aggressive trainer produced a network that drove recklessly and tended to cut off other cars (Figure 4).

## Discussion

One of the strengths of expert systems based on ANS is that the use of input data in the decision making and control process does not have to be specified. The network adapts its internal weights to conform to input/output correlations it discovers. It is important, however, that data used by the human expert is also available to the network. The different processing of sensor data for man and network may have important consequences, key information may be presented to the man but not the machine.

This difference in data processing is particularly worrisome for image data where human ability to extract detail is vastly superior to our automatic image processing capabilities. Though we would not require an image processing system to understand images, it would have to extract relevant information from cluttered backgrounds. Until we have sufficiently sophisticated algorithms or networks to do this, our efforts at constructing expert systems which handle image data are handicapped.

Scaling input data to the unitary order of magnitude is important for training stability. This is evident from equations (1) and (2). The sigmoid transfer function ranges from 0.1 to 0.9 in approximately four units, that is, over an $O(1)$ domain. If system response must change in reaction to a large, $O(n)$ swing of a given input parameter, the weight associated with that input will be trained toward an $O(n^{-1})$ magnitude. On the other hand, if the same system responds to an input whose range is $O(1)$, its associated weight will also be $O(1)$. The weight adjustment equation does not recognize differences in weight magnitude, therefore relatively small weights will undergo wild magnitude adjustments and converge weakly. On the other hand, if all input parameters are of the same magnitude their associated weights will reflect this and the training constant can be adjusted for gentle weight convergence. Because the output of hidden units are constrained between zero and one, $O(1)$ is a good target range for input parameters. Both the hyperbolic tangent and logarithmic functions are useful for scaling wide ranging inputs. A useful form of the latter is

$$
x' = \begin{cases}
\beta[1+\ln(x/\alpha)] & \text{if } \alpha < x, \\
\beta x/\alpha & \text{if } -\alpha \leq x \leq \alpha, \\
-\beta[1+\ln(-x/\alpha)] & \text{if } x < -\alpha,
\end{cases}
\tag{4}
$$

where $\alpha > 0$ and defines the limits of the intermediate linear section, and $\beta$ is a scaling factor. This symmetric logarithmic function is continuous in its first derivative, and useful when network behavior should change slowly as a parameter increases without bound. On the other hand, if the system should approach a limiting behavior, the tanh function is appropriate.

Weight adaptation is also complicated by relaxing the common practice of restricting interconnections to adjacent layers. Equation (3) shows that the calculated error for a hidden layer-given comparable weights, fanouts and output errors-will be one quarter or less than that of the

output layer. This is caused by the slope factor, $o_i(1-o_i)$. The difference in error magnitudes is not noticeable in networks restricted to adjacent layer interconnectivity. But when this constraint is released the effect of errors originating directly from an output unit has $4^d$ times the magnitude and effect of an error originating from a hidden unit removed $d$ layers from the output layer. Compared to the corrections arising from the output units, those from the hidden units have little influence on weight adjustment, and the power of a multilayer structure is weakened. The system will train if we restrict connections to adjacent layers, but it trains slowly. To compensate for this effect we attenuate the error magnitudes originating from the output layer by the above factor. This heuristic procedure works well and facilitates smooth learning.

Though we have made progress in real-time learning systems using GDR, compared to humans-who can learn from a single data presentation-they remain relatively sluggish in learning and response rates. We are interested in improvements of the GDR algorithm or alternative architectures that facilitate one-shot or rapid learning. In the latter case we are considering least squares restoration techniques[4] and Grossberg and Carpenter's adaptive resonance models[3,5].

The construction of automated expert systems by observation of human personnel is attractive because of its efficient use of the expert's time and effort. Though the classic AI approach of rule base inference is applicable when such rules are clear cut and well organized, too often a human expert can not put his decision making process in words or specify the values of parameters that influence him. The attraction of ANS based systems is that imitations of expert behavior emerge as a natural consequence of their training.

### References

1) D. E. Rumelhart, G. E. Hinton, and R. J. Williams, "Learning Internal Representations by Error Propagation," in *Parallel Distributed Processing: Explorations in the Microstructure of Cognition, Vol. I*, D. E. Rumelhart and J. L. McClelland (Eds.), chap. 8, (1986), Bradford Books/MIT Press, Cambridge

2) S. Grossberg, *Studies of Mind and Brain*, (1982), Reidel, Boston

3) A. Barto and R. Sutton, "Landmark Learning: An Illustration of Associative Search," *Biological Cybernetics*, 42, (1981), p. 1

4) A. Rosenfeld and A. Kak, *Digital Picture Processing, Vol. 1*, chap. 7, (1982), Academic Press, New York

5) G. A. Carpenter and S. Grossberg, "A Massively Parallel Architecture for a Self-organizing Neural Pattern Recognition Machine," *Computer Vision, Graphics and Image Processing*, 37, (1987), p.54
